# Bayesian active learning with localized priors for fast receptive field characterization

**Mijung Park**
Electrical and Computer Engineering
The University of Texas at Austin
mjpark@mail.utexas.edu

**Jonathan W. Pillow**
Center For Perceptual Systems
The University of Texas at Austin
pillow@mail.utexas.edu

## Abstract

Active learning methods can dramatically improve the yield of neurophysiology experiments by adaptively selecting stimuli to probe a neuron's receptive field (RF). Bayesian active learning methods specify a posterior distribution over the RF given the data collected so far in the experiment, and select a stimulus on each time step that maximally reduces posterior uncertainty. However, existing methods tend to employ simple Gaussian priors over the RF and do not exploit uncertainty at the level of hyperparameters. Incorporating this uncertainty can substantially speed up active learning, particularly when RFs are smooth, sparse, or local in space and time. Here we describe a novel framework for active learning under hierarchical, conditionally Gaussian priors. Our algorithm uses sequential Markov Chain Monte Carlo sampling ("particle filtering" with MCMC) to construct a mixture-of-Gaussians representation of the RF posterior, and selects optimal stimuli using an approximate infomax criterion. The core elements of this algorithm are parallelizable, making it computationally efficient for real-time experiments. We apply our algorithm to simulated and real neural data, and show that it can provide highly accurate receptive field estimates from very limited data, even with a small number of hyperparameter samples.

## 1   Introduction

Neurophysiology experiments are costly and time-consuming. Data are limited by an animal's willingness to perform a task (in awake experiments) and the difficulty of maintaining stable neural recordings. This motivates the use of active learning, known in statistics as "optimal experimental design", to improve experiments using adaptive stimulus selection in closed-loop experiments. These methods are especially powerful for models with many parameters, where traditional methods typically require large amounts of data.

In Bayesian active learning, the basic idea is to define a statistical model of the neural response, then carry out experiments to efficiently characterize the model parameters [1–6]. (See Fig. 1A). Typically, this begins with a (weakly- or non-informative) prior distribution, which expresses our uncertainty about these parameters before the start of the experiment. Then, recorded data (i.e., stimulus-response pairs) provide likelihood terms that we combine with the prior to obtain a posterior distribution. This posterior reflects our beliefs about the parameters given the data collected so far in the experiment. We then select a stimulus for the next trial that maximizes some measure of utility (e.g., expected reduction in entropy, mean-squared error, classification error, etc.), integrated with respect to the current posterior.

In this paper, we focus on the problem of receptive field (RF) characterization from extracellularly recorded spike train data. The receptive field is a linear filter that describes how the neuron integrates its input (e.g., light) over space and time; it can be equated with the linear term in a generalized linear

model (GLM) of the neural response [7]. Typically, RFs are high-dimensional (with 10s to 100s of parameters, depending on the choice of input domain), making them an attractive target for active learning methods. Our paper builds on prior work from Lewi *et al* [6], a seminal paper that describes active learning for RFs under a conditionally Poisson point process model.

Here we show that a sophisticated choice of prior distribution can lead to substantial improvements in active learning. Specifically, we develop a method for learning under a class of hierarchical, conditionally Gaussian priors that have been recently developed for RF estimation [8,9]. These priors flexibly encode a preference for smooth, sparse, and/or localized structure, which are common features of real neural RFs. In fixed datasets ("passive learning"), the associated estimators give substantial improvements over both maximum likelihood and standard lasso/ridge-regression shrinkage estimators, but they have not yet been incorporated into frameworks for active learning.

Active learning with a non-Gaussian prior poses several major challenges, however, since the posterior is non-Gaussian, and requisite posterior expectations are much harder to compute. We address these challenges by exploiting a conditionally Gaussian representation of the prior (and posterior) using sampling at the level of the hyperparameters. We demonstrate our method using the *Automatic Locality Determination* (ALD) prior introduced in [9], where hyperparameters control the locality of the RF in space-time and frequency. The resulting algorithm outperforms previous active learning methods on real and simulated neural data, even under various forms of model mismatch.

The paper is organized as follows. In Sec. 2, we formally define the Bayesian active learning problem and review the algorithm of [6], to which we will compare our results. In Sec. 3, we describe a hierarchical response model, and in Sec. 4 describe the localized RF prior that we will employ for active learning. In Sec. 5, we describe a new active learning method for conditionally Gaussian priors. In Sec. 6, we show results of simulated experiments with simulated and real neural data.

## 2 Bayesian active learning

Bayesian active learning (or "experimental design") provides a model-based framework for selecting optimal stimuli or experiments. A Bayesian active learning method has three basic ingredients: (1) an observation model (likelihood) $p(y|\mathbf{x}, \mathbf{k})$, specifying the conditional probability of a scalar response $y$ given vector stimulus $\mathbf{x}$ and parameter vector $\mathbf{k}$; (2) a prior $p(\mathbf{k})$ over the parameters of interest; and (3) a loss or utility function $U$, which characterizes the desirability of a stimulus-response pair $(\mathbf{x}, y)$ under the current posterior over $\mathbf{k}$. The optimal stimulus $\mathbf{x}$ is the one that maximizes the expected utility $\mathbb{E}_{y|\mathbf{x}}[U(\mathbf{x}, y)]$, meaning the utility averaged over the distribution of (as yet) unobserved $y|\mathbf{x}$.

One popular choice of utility function is the mutual information between $(\mathbf{x}, y)$ and the parameters $\mathbf{k}$. This is commonly known as information-theoretic or *infomax* learning [10]. It is equivalent to picking the stimulus on each trial that minimizes the expected posterior entropy.

Let $\mathcal{D}_t = \{x_i, y_i\}_{i=1}^t$ denote the data collected up to time step $t$ in the experiment. Under infomax learning, the optimal stimulus at time step $t + 1$ is:

$$\mathbf{x}_{t+1} \quad = \quad \arg\max_{\mathbf{x}} \mathbb{E}_{y|\mathbf{x}, \mathcal{D}_t}[I(y, \mathbf{k}|\mathbf{x}, \mathcal{D}_t)] = \arg\min_{\mathbf{x}} \mathbb{E}_{y|\mathbf{x}, \mathcal{D}_t}[H(\mathbf{k}|\mathbf{x}, y, \mathcal{D}_t)], \quad (1)$$

where $H(\mathbf{k}|\mathbf{x}, y, \mathcal{D}_t) = -\int p(\mathbf{k}|\mathbf{x}, y, \mathcal{D}_t) \log p(\mathbf{k}|\mathbf{x}, y, \mathcal{D}_t) d\mathbf{k}$ denotes the posterior entropy of $\mathbf{k}$, and $p(y|\mathbf{x}, \mathcal{D}_t) = \int p(y|\mathbf{x}, \mathbf{k}) p(\mathbf{k}|\mathcal{D}_t) d\mathbf{k}$ is the predictive distribution over response $y$ given stimulus $\mathbf{x}$ and data $\mathcal{D}_t$. The mutual information provided by $(y, \mathbf{x})$ about $\mathbf{k}$, denoted by $I(y, \mathbf{k}|\mathbf{x}, \mathcal{D}_t)$, is simply the difference between the prior and posterior entropy.

### 2.1 Method of Lewi, Butera & Paninski 2009

Lewi *et al* [6] developed a Bayesian active learning framework for RF characterization in closed-loop neurophysiology experiments, which we henceforth refer to as "Lewi-09". This method employs a conditionally Poisson generalized linear model (GLM) of the neural spike response:

$$\begin{aligned} \lambda_t &= g(\mathbf{k}^\top \mathbf{x}_t) \\ y_t &\sim \text{Poiss}(\lambda_t), \end{aligned} \quad (2)$$

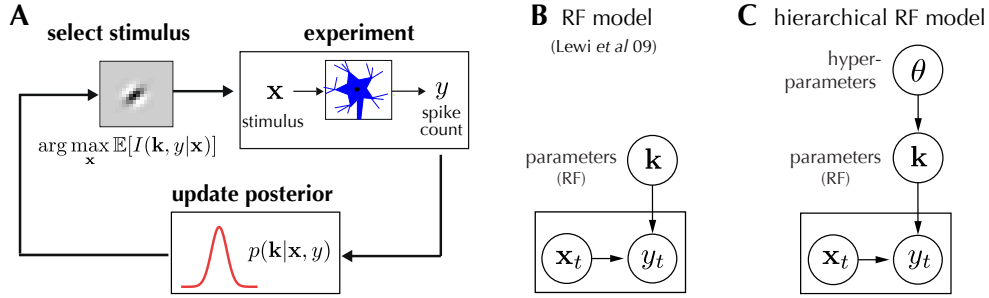

Figure 1: **(A)** Schematic of Bayesian active learning for neurophysiology experiments. For each presented stimulus $\mathbf{x}$ and recorded response $y$ (upper right), we update the posterior over receptive field $\mathbf{k}$ (bottom), then select the stimulus that maximizes expected information gain (upper left). **(B)** Graphical model for the non-hierarchical RF model used by Lewi-09. It assumes a Gaussian prior $p(\mathbf{k})$ and Poisson likelihood $p(y_t|\mathbf{x}_t, \mathbf{k})$. **(C)** Graphical model for the hierarchical RF model used here, with a hyper-prior $p_\theta(\theta)$ over hyper-parameters and conditionally Gaussian prior $p(\mathbf{k}|\theta)$ over the RF. For simplicity and speed, we assume a Gaussian likelihood for $p(y_t|\mathbf{x}_t, \mathbf{k})$, though all examples in the manuscript involved real neural data or simulations from a Poisson GLM.

where $g$ is a nonlinear function that ensures non-negative spike rate $\lambda_t$.

The Lewi-09 method assumes a Gaussian prior over $\mathbf{k}$, which leads to a (non-Gaussian) posterior given by the product of Poisson likelihood and Gaussian prior. (See Fig. 1B). Neither the predictive distribution $p(y|\mathbf{x}, \mathcal{D}_t)$ nor the posterior entropy $H(\mathbf{k}|\mathbf{x}, y, \mathcal{D}_t)$ can be computed in closed form. However, the log-concavity of the posterior (guaranteed for suitable choice of $g$ [11]) motivates a tractable and accurate Gaussian approximation to the posterior, which provides a concise analytic formula for posterior entropy [12, 13].

The key contributions of Lewi-09 include fast methods for updating the Gaussian approximation to the posterior and for selecting the stimulus (subject to a maximum-power constraint) that maximizes expected information gain. The Lewi-09 algorithm yields substantial improvement in characterization performance relative to randomized *iid* (e.g., "white noise") stimulus selection. Below, we will benchmark the performance of our method against this algorithm.

## 3 Hierarchical RF models

Here we seek to extend the work of Lewi *et al* to incorporate non-Gaussian priors in a hierarchical receptive field model. (See Fig. 1C). Intuitively, a good prior can improve active learning by reducing the prior entropy, i.e., the effective size of the parameter space to be searched. The drawback of more sophisticated priors is that they may complicate the problem of computing and optimizing the posterior expectations needed for active learning.

To focus more straightforwardly on the role of the prior distribution, we employ a simple linear-Gaussian model of the neural response:

$$y_t = \mathbf{k}^\top \mathbf{x}_t + \epsilon_t, \quad \epsilon_t \sim \mathcal{N}(0, \sigma^2), \tag{3}$$

where $\epsilon_t$ is *iid* zero-mean Gaussian noise with variance $\sigma^2$. We then place a hierarchical, conditionally Gaussian prior on $\mathbf{k}$:

$$\mathbf{k} \mid \theta \quad \sim \quad \mathcal{N}(0, C_\theta) \tag{4}$$
$$\theta \quad \sim \quad p_\theta, \tag{5}$$

where $C_\theta$ is a prior covariance matrix that depends on hyperparameters $\theta$. These hyperparameters in turn have a hyper-prior $p_\theta$. We will specify the functional form of $C_\theta$ in the next section.

In this setup, the effective prior over $\mathbf{k}$ is a mixture-of-Gaussians, obtained by marginalizing over $\theta$:

$$p(\mathbf{k}) = \int p(\mathbf{k}|\theta)p(\theta)d\theta = \int \mathcal{N}(0, C_\theta)\, p_\theta(\theta)d\theta. \tag{6}$$

Given data $X = (\mathbf{x}_1, \ldots, \mathbf{x}_t)^\top$ and $Y = (y_1, \ldots, y_t)^\top$, the posterior also takes the form of a mixture-of-Gaussians:

$$p(\mathbf{k}|X, Y) = \int p(\mathbf{k}|X, Y, \theta) p(\theta|X, Y) d\theta \tag{7}$$

where the conditional posterior given $\theta$ is the Gaussian

$$p(\mathbf{k}|X, Y, \theta) = \mathcal{N}(\mu_\theta, \Lambda_\theta), \qquad \mu_\theta = \tfrac{1}{\sigma^2} \Lambda_\theta X^\top Y, \quad \Lambda_\theta = (\tfrac{1}{\sigma^2} X^\top X + C_\theta^{-1})^{-1}, \tag{8}$$

and the mixing weights are given by the marginal posterior,

$$p(\theta|X, Y) \propto p(Y|X, \theta) p_\theta(\theta), \tag{9}$$

which we will only need up to a constant of proportionality. The marginal likelihood or *evidence* $p(Y|X, \theta)$ is the marginal probability of the data given the hyperparameters, and has a closed form for the linear Gaussian model:

$$p(Y|X, \theta) = \frac{|2\pi\Lambda_\theta|^{\frac{1}{2}}}{|2\pi\sigma^2 I|^{\frac{1}{2}} |2\pi C_\theta|^{\frac{1}{2}}} \exp\left[\tfrac{1}{2}\left(\mu_\theta^\top \Lambda_\theta^{-1} \mu_\theta - m^\top L^{-1} m\right)\right], \tag{10}$$

where $L = \sigma^2 (X^\top X)^{-1}$ and $m = \tfrac{1}{\sigma^2} L X^\top Y$.

Several authors have pointed out that active learning confers no benefit over fixed-design experiments in linear-Gaussian models with Gaussian priors, due to the fact that the posterior covariance is response-independent [1, 6]. That is, an optimal design (one that minimizes the final posterior entropy) can be planned out entirely in advance of the experiment. However, this does not hold for linear-Gaussian models with *non-Gaussian* priors, such as those considered here. The posterior distribution in such models *is* data-dependent via the marginal posterior's dependence on $Y$ (eq. 9). Thus, active learning is warranted even for linear-Gaussian responses, as we will demonstrate empirically below.

## 4  Automatic Locality Determination (ALD) prior

In this paper, we employ a flexible RF model underlying the so-called *automatic locality determination* (ALD) estimator [9].[1] The key justification for the ALD prior is the observation that most neural RFs tend to be localized in both space-time and spatio-temporal frequency. Locality in space-time refers to the fact that (e.g., visual) neurons integrate input over a limited domain in time and space; locality in frequency refers to the band-pass (or smooth / low pass) character of most neural RFs. The ALD prior encodes these tendencies in the parametric form of the covariance matrix $C_\theta$, where hyperparameters $\theta$ control the support of both the RF and its Fourier transform.

The hyperparameters for the ALD prior are $\theta = (\rho, \nu_s, \nu_f, M_s, M_f)^\top$, where $\rho$ is a "ridge" parameter that determines the overall amplitude of the covariance; $\nu_s$ and $\nu_f$ are length-D vectors that specify the center of the RF support in space-time and frequency, respectively (where $D$ is the degree of the RF tensor[2]); and $M_s$ and $M_f$ are $D \times D$ positive definite matrices that describe an elliptical (Gaussian) region of support for the RF in space-time and frequency, respectively. In practice, we will also include the additive noise variance $\sigma^2$ (eq. 3) as a hyperparameter, since it plays a similar role to $C$ in determining the posterior and evidence. Thus, for the ($D = 2$) examples considered here, there are 12 hyperparameters, including scalars $\sigma^2$ and $\rho$, two hyperparameters each for $\nu_s$ and $\nu_f$, and three each for symmetric matrices $M_s$ and $M_f$.

Note that although the conditional ALD prior over $\mathbf{k}|\theta$ assigns high prior probability to smooth and sparse RFs for some settings of $\theta$, for other settings (i.e., where $M_s$ and $M_f$ describe elliptical regions large enough to cover the entire RF) the conditional prior corresponds to a simple ridge prior and imposes no such structure. We place a flat prior over $\theta$ so that no strong prior beliefs about spatial locality or bandpass frequency characteristics are imposed *a priori*. However, as data from a neuron with a truly localized RF accumulates, the support of the marginal posterior $p(\theta|\mathcal{D}_t)$ shrinks down on regions that favor a localized RF, shrinking the posterior entropy over $\mathbf{k}$ far more quickly than is achievable with methods based on Gaussian priors.

# 5 Bayesian active learning with ALD

To perform active learning under the ALD model, we need two basic ingredients: (1) an efficient method for representing and updating the posterior $p(\mathbf{k}|\mathcal{D}_t)$ as data come in during the experiment; and (2) an efficient algorithm for computing and maximizing the expected information gain given a stimulus $\mathbf{x}$. We will describe each of these in turn below.

## 5.1 Posterior updating via sequential Markov Chain Monte Carlo

To represent the ALD posterior over $\mathbf{k}$ given data, we will rely on the conditionally Gaussian representation of the posterior (eq. 7) using particles $\{\theta_i\}_{i=1,...,N}$ sampled from the marginal posterior, $\theta_i \sim P(\theta|\mathcal{D}_t)$ (eq. 9). The posterior will then be approximated as:

$$p(\mathbf{k}|\mathcal{D}_t) \approx \frac{1}{N} \sum_i p(\mathbf{k}|\mathcal{D}_t, \theta_i), \tag{11}$$

where each distribution $p(\mathbf{k}|\mathcal{D}_t, \theta_i)$ is Gaussian with $\theta_i$-dependent mean and covariance (eq. 8).

Markov Chain Monte Carlo (MCMC) is a popular method for sampling from distributions known only up to a normalizing constant. In cases where the target distribution evolves over time by accumulating more data, however, MCMC samplers are often impractical due to the time required for convergence (i.e., "burning in"). To reduce the computational burden, we use a sequential sampling algorithm to update the samples of the hyperparameters at each time step, based on the samples drawn at the previous time step. The main idea of our algorithm is adopted from the *resample-move particle filter*, which involves generating initial particles; resampling particles according to incoming data; then performing MCMC moves to avoid degeneracy in particles [14]. The details are as follows.

**Initialization:** On the first time step, generate initial hyperparameter samples $\{\theta_i\}$ from the hyperprior $p_\theta$, which we take to be flat over a broad range in $\theta$.

**Resampling:** Given a new stimulus/response pair $\{\mathbf{x}, y\}$ at time $t$, resample the existing particles according to the *importance weights*:

$$p(y_t|\theta_i^{(t)}, \mathcal{D}_{t-1}, \mathbf{x}_t) = \mathcal{N}(y_t|\mu_i^\top \mathbf{x}_t, \ \mathbf{x}_t^\top \Lambda_i \mathbf{x}_t + \sigma_i^2), \tag{12}$$

where $(\mu_i, \Lambda_i)$ denote the mean and covariance of the Gaussian component attached to particle $\theta_i$, This ensures the posterior evolves according to:

$$p(\theta_i^{(t)}|\mathcal{D}_t) \propto p(y_t|\theta_i^{(t)}, \mathcal{D}_{t-1}, \mathbf{x}_t)p(\theta_i^{(t)}|\mathcal{D}_{t-1}). \tag{13}$$

**MCMC Move:** Propagate particles via Metropolis Hastings (MH), with multivariate Gaussian proposals centered on the current particle $\theta_i$ of the Markov chain: $\theta^* \sim \mathcal{N}(\theta_i, \Gamma)$, where $\Gamma$ is a diagonal matrix with diagonal entries given by the variance of the particles at the end of time step $t-1$. Accept the proposal with probability $\min(1, \alpha)$, where $\alpha = \frac{q(\theta^*)}{q(\theta_i)}$, with $q(\theta_i) = p(\theta_i|\mathcal{D}_t)$. Repeat MCMC moves until computational or time budget has expired.

The main bottleneck of this scheme is the updating of conditional posterior mean $\mu_i$ and covariance $\Lambda_i$ for each particle $\theta_i$, since this requires inversion of a $d \times d$ matrix. (Note that, unlike Lewi-09, these are not rank-one updates due to the fact that $C_{\theta_i}$ changes after each $\theta_i$ move). This cost is independent of the amount of data, linear in the number of particles, and scales as $O(d^3)$ in RF dimensionality $d$. However, particle updates can be performed efficiently in parallel on GPUs or machines with multi-core processors, since the particles do not interact except for stimulus selection, which we describe below.

## 5.2 Optimal Stimulus Selection

Given the posterior over $\mathbf{k}$ at time $t$, represented by a mixture of Gaussians attached to particles $\{\theta_i\}$ sampled from the marginal posterior, our task is to determine the maximally informative stimulus to present at time $t + 1$. Although the entropy of a mixture-of-Gaussians has no analytic form, we can

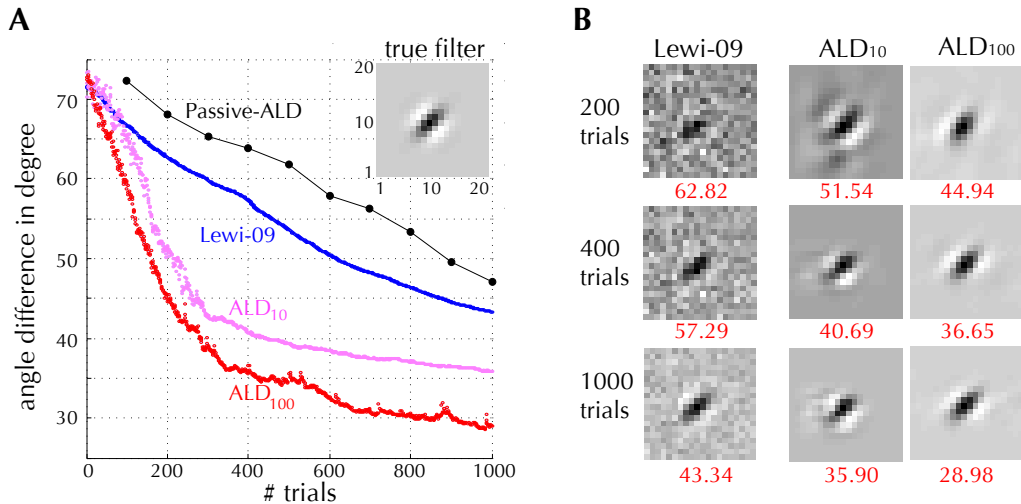

Figure 2: Simulated experiment. **(A)** Angular error in estimates of a simulated RF ($20 \times 20$ pixels, shown in inset) vs. number of stimuli, for Lewi-09 method (blue), the ALD-based active learning method using 10 (pink) or 100 (red) particles, and the ALD-based passive learning method (black). True responses were simulated from a Poisson-GLM neuron. Traces show average over 20 independent repetitions. **(B)** RF estimates obtained by each method after 200, 400, and 1000 trials. Red numbers below indicate angular error (deg).

compute the exact posterior covariance via the formula:

$$\tilde{\Lambda}_t = \frac{1}{N} \sum_{i=1}^{N} \left( \Lambda_i + \mu_i \mu_i^\top \right) - \tilde{\mu} \tilde{\mu}^\top, \tag{14}$$

where $\tilde{\mu}_t = \frac{1}{N} \sum \mu_i$ is the full posterior mean. This leads to an upper bound on posterior entropy, since a Gaussian is the maximum-entropy distribution for fixed covariance. We then take the next stimulus to be the maximum-variance eigenvector of the posterior covariance, which is the most informative stimulus under a Gaussian posterior and Gaussian noise model, subject to a power constraint on stimuli [6].

Although this selection criterion is heuristic, since it is not guaranteed to maximize mutual information under the true posterior, it is intuitively reasonable since it selects the stimulus direction along which the current posterior is maximally uncertain. Conceptually, directions of large posterior variance can arise in two different ways: (1) directions of large variance for all covariances $\Lambda_i$, meaning that all particles assign high posterior uncertainty over $\mathbf{k}|\mathcal{D}_t$ in the direction of $\mathbf{x}$; or (2) directions in which the means $\mu_i$ are highly dispersed, meaning the particles disagree about the mean of $\mathbf{k}|\mathcal{D}_t$ in the direction of $\mathbf{x}$. In either scenario, selecting a stimulus proportional to the dominant eigenvector is heuristically justified by the fact that it will reduce collective uncertainty in particle covariances or cause particle means to converge by narrowing of the marginal posterior. We show that the method performs well in practice for both real and simulated data (Section 6). We summarize the complete method in Algorithm 1.

---

**Algorithm 1** Sequential active learning under conditionally Gaussian models

---

Given particles $\{\theta_i\}$ from $p(\theta|\mathcal{D}_t)$, which define the posterior as $P(\mathbf{k}|\mathcal{D}_t) = \sum_i \mathcal{N}(\mu_i, \Lambda_i)$,
**1**. Compute the posterior covariance $\tilde{\Lambda}_t$ from $\{(\mu_i, \Lambda_i)\}$ (eq. 14).
**2**. Select optimal stimulus $\mathbf{x}_{t+1}$ as the maximal eigenvector of $\tilde{\Lambda}_t$
**3**. Measure response $y_{t+1}$.
**4**. Resample particles $\{\theta_i\}$ with the weights $\{\mathcal{N}(y_{t+1}|\mu_i^\top \mathbf{x}_{t+1}, \mathbf{x}_{t+1}^\top \Lambda_i \mathbf{x}_{t+1} + \sigma_i^2)\}$.
**5**. Perform MH sampling of $p(\theta|\mathcal{D}_{t+1})$, starting from resampled particles.
**repeat**

---

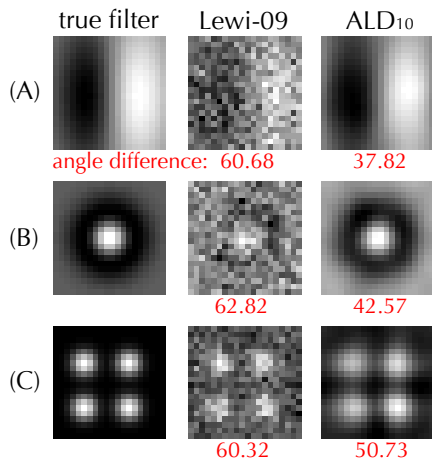

true filter     Lewi-09     $ALD_{10}$

(A)

angle difference:  60.68      37.82

(B)

62.82      42.57

(C)

60.32      50.73

Figure 3: Additional simulated examples comparing Lewi-09 and ALD-based active learning. Responses were simulated from a GLM-Poisson model with three different true 400-pixel RFs (left column): **(A)** a Gabor filter (shown previously in [6]); **(B)**: a center-surround RF, typical in retinal ganglion cells; **(C)**: a relatively non-localized grid-cell RF. Middle and right columns show RF estimates after 400 trials of active learning under each method, with average angular error (over independent 20 repeats) shown beneath in red.

## 6 Results

**Simulated Data**: We tested the performance of our algorithm using data simulated from a Poisson-GLM neuron with a $20 \times 20$ pixel Gabor filter and an exponential nonlinearity (See Fig. 2). This is the response model assumed by the Lewi-09 method, and therefore substantially mismatched to the linear-Gaussian model assumed by our method.

For the Lewi-09 method, we used a diagonal prior covariance with amplitude set by maximizing marginal likelihood for a small dataset. We compared two versions of the ALD-based algorithm (with 10 and 100 hyperparameter particles, respectively) to examine the relationship between performance and fidelity of the posterior representation. To quantify the performance, we used the angular difference (in degrees) between the true and estimated RF.

Fig 2A shows the angular difference between the true RF and estimates obtained by Lewi-09 and the ALD-based method, as a function of the number of trials. The ALD estimate exhibits more rapid convergence, and performs noticeably better with 100 than with 10 particles ($ALD_{100}$ vs. $ALD_{10}$), indicating that accurately preserving uncertainty over the hyperparameters is beneficial to performance. We also show the performance of ALD inference under passive learning (*iid* random stimulus selection), which indicates that the improvement in our method is not simply due to the use of an improved RF estimator. Fig 2B shows the estimates obtained by each method after 200, 400, and 1000 trials. Note that the estimate with 100 hyperparameter samples is almost indistinguishable from the true filter after 200 trials, which is substantially lower than the dimensionality of the filter itself ($d = 400$).

Fig. 3 shows a performance comparison using three additional 2-dimensional receptive fields, to show that performance improves across a variety of different RF shapes. The filters included: (A) a gabor filter similar to that used in [6]; (B) a retina-like center-surround receptive field; (C) a grid-cell receptive field with multiple modes. As before, noisy responses were simulated from a Poisson-GLM. For the grid-cell example, these filter is not strongly localized in space, yet the ALD-based estimate substantially outperforms Lewi-09 due to its sensitivity to localized components in frequency. Thus, ALD-based method converges more quickly despite the mismatch between the model used to simulate data and the model assumed for active learning.

**Neural Data**: We also tested our method with an off-line analysis of real neural data from a simple cell recorded in primate V1 (published in [15]). The stimulus consisted of 1D spatiotemporal white noise ("flickering bars"), with 16 spatial bars on each frame, aligned with the cell's preferred orientation. We took the RF to have 16 time bins, resulting in a 256-dimensional parameter space for the RF. We performed simulated active learning by extracting the raw stimuli from 46 minutes of experimental data. On each trial, we then computed the expected information gain from presenting each of these stimuli (blind to neuron's actual response to each stimulus). We used ALD-based active learning with 10 hyperparameter particles, and examined performance of both algorithms for 960 trials (selecting from $\approx 276{,}000$ possible stimuli on each trial).

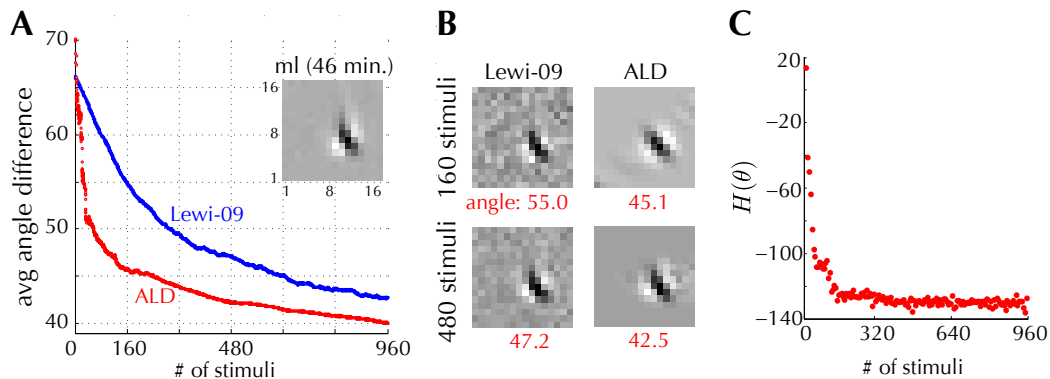

Figure 4: Comparison of active learning methods in a simulated experiment with real neural data from a primate V1 simple cell. (Original data recorded in response to white noise "flickering bars" stimuli, see [15]). **(A)**: Average angular difference between the MLE (inset, computed from an entire 46-minute dataset) and the estimates obtained by active learning, as a function of the amount of data. We simulated active learning via an offline analysis of the fixed dataset, where methods had access to possible stimuli but not responses. **(B)**: RF estimates after 10 and 30 seconds of data. Note that the ALD-based estimate has smaller error with 10 seconds of data than Lewi-09 with 30 seconds of data. **(C)**: Average entropy of hyperparameter particles as a function of $t$, showing rapid narrowing of marginal posterior.

Fig 4A shows the average angular difference between the maximum likelihood estimate (computed with the entire dataset) and the estimate obtained by each active learning method, as a function of the number of stimuli. The ALD-based method reduces the angular difference by 45 degrees with only 160 stimuli, although the filter dimensionality of the RF for this example is 256. The Lewi-09 method requires four times more data to achieve the same accuracy. Fig 4B shows estimates after 160 and 480 stimuli. We also examined the average entropy of the hyperparameter particles as a function of the amount of data used. Fig. 4C shows that the entropy of the marginal posterior over hyperparameters falls rapidly during the first 150 trials of active learning.

The main bottleneck of the algorithm is eigendecomposition of the posterior covariance $\tilde{\Lambda}$, which took 30ms for a $256 \times 256$ matrix on a $2 \times 2.66$ GHz Quad-Core Intel Xeon Mac Pro. Updating importance weights and resampling 10 particles took 4ms, and a single step of MH resampling for each particle took 5ms. In total, it took <60 ms to compute the optimal stimulus in each trial using a non-optimized implementation of our algorithm, indicating that our methods should be fast enough for use in real-time neurophysiology experiments.

## 7 Discussion

We have developed a Bayesian active learning method for neural RFs under hierarchical response models with conditionally Gaussian priors. To take account of uncertainty at the level of hyperparameters, we developed an approximate information-theoretic criterion for selecting optimal stimuli under a mixture-of-Gaussians posterior. We applied this framework using a prior designed to capture smooth and localized RF structure. The resulting method showed clear advantages over traditional designs that do not exploit structured prior knowledge. We have contrasted our method with that of Lewi *et al* [6], which employs a more flexible and accurate model of the neural response, but a less flexible model of the RF prior. A natural future direction therefore will be to combine the Poisson-GLM likelihood and ALD prior, which will combine the benefits of a more accurate neural response model and a flexible (low-entropy) prior for neural receptive fields, while incurring only a small increase in computational cost.

## Acknowledgments

We thank N. C. Rust and J. A. Movshon for V1 data, and several anonymous reviewers for helpful advice on the original manuscript. This work was supported by a Sloan Research Fellowship, McKnight Scholar's Award, and NSF CAREER Award IIS-1150186 (JP).

## Footnotes

[1]"Automatic" refers to the fact that in [9], the model was used for empirical Bayes inference, i.e., MAP inference after maximizing the evidence for $\theta$. Here, we consider perform fully Bayesian inference under the associated model.

[2]e.g., a *space×space×time* RF has degree $D = 3$.

## References

[1] D. J. C. MacKay. Information-based objective functions for active data selection. *Neural Computation*, 4(4):590–604, 1992.

[2] K. Chaloner and I. Verdinelli. Bayesian experimental design: a review. *Statistical Science*, 10:273–304, 1995.

[3] D. A. Cohn, Z. Ghahramani, and M. I. Jordan. Active learning with statistical models. *J. Artif. Intell. Res. (JAIR)*, 4:129–145, 1996.

[4] A. Watson and D. Pelli. QUEST: a Bayesian adaptive psychophysical method. *Perception and Psychophysics*, 33:113–120, 1983.

[5] L. Paninski. Asymptotic theory of information-theoretic experimental design. *Neural Computation*, 17(7):1480–1507, 2005.

[6] J. Lewi, R. Butera, and L. Paninski. Sequential optimal design of neurophysiology experiments. *Neural Computation*, 21(3):619–687, 2009.

[7] W. Truccolo, U. T. Eden, M. R. Fellows, J. P. Donoghue, and E. N. Brown. A point process framework for relating neural spiking activity to spiking history, neural ensemble and extrinsic covariate effects. *J. Neurophysiol*, 93(2):1074–1089, 2005.

[8] M. Sahani and J. Linden. Evidence optimization techniques for estimating stimulus-response functions. *NIPS*, 15, 2003.

[9] M. Park and J. W. Pillow. Receptive field inference with localized priors. *PLoS Comput Biol*, 7(10):e1002219, 2011.

[10] N. Houlsby, F. Huszar, Z. Ghahramani, and M. Lengyel. Bayesian active learning for classification and preference learning. *CoRR*, abs/1112.5745, 2011.

[11] L. Paninski. Maximum likelihood estimation of cascade point-process neural encoding models. *Network: Computation in Neural Systems*, 15:243–262, 2004.

[12] R. Kass and A. Raftery. Bayes factors. *Journal of the American Statistical Association*, 90:773–795, 1995.

[13] J. W. Pillow, Y. Ahmadian, and L. Paninski. Model-based decoding, information estimation, and change-point detection techniques for multineuron spike trains. *Neural Comput*, 23(1):1–45, Jan 2011.

[14] W. R. Gilks and C. Berzuini. Following a moving target – monte carlo inference for dynamic bayesian models. *Journal of the Royal Statistical Society: Series B (Statistical Methodology)*, 63(1):127–146, 2001.

[15] N. C. Rust, Schwartz O., J. A. Movshon, and Simoncelli E.P. Spatiotemporal elements of macaque v1 receptive fields. *Neuron*, 46(6):945–956, 2005.

